# Improving Committee Diagnosis with Resampling Techniques

**Bambang Parmanto**
Department of Information Science
University of Pittsburgh
Pittsburgh, PA 15260
*parmanto@lis.pitt.edu*

**Paul W. Munro**
Department of Information Science
University of Pittsburgh
Pittsburgh, PA 15260
*munro@lis.pitt.edu*

**Howard R. Doyle**
Pittsburgh Transplantation Institute
3601 Fifth Ave, Pittsburgh, PA 15213
*doyle@vesalius.tzs.med.pitt.edu*

## Abstract

Central to the performance improvement of a committee relative to individual networks is the error correlation between networks in the committee. We investigated methods of achieving error independence between the networks by training the networks with different resampling sets from the original training set. The methods were tested on the sinwave artificial task and the real-world problems of hepatoma (liver cancer) and breast cancer diagnoses.

## 1  INTRODUCTION

The idea of a neural net committee is to combine several neural net predictors to perform collective decision making, instead of using a single network (Perrone, 1993). The potential of a committee in improving classification performance has been well documented. Central to this improvement is the extent to which the errors tend to coincide. Committee errors occur where the misclassification sets of individual networks overlap. On the one hand, if all errors of committee members coincide, using a committee does not improve performance. On the other hand, if errors do not coincide, performance of the committee dramatically increases and asymptotically approaches perfect performance. Therefore, it is beneficial to make the errors among the networks in the committee less correlated in order to improve the committee performance.

One way of making the networks less correlated is to train them with different sets of data. Decreasing the error correlation by training members of the committee using different sets of data is intuitively appealing. Networks trained with different data sets have a higher probability of generalizing differently and tend to make errors in different places in the problem space.

The idea is to split the data used in the training into several sets. The sets are not necessarily mutually exclusive, they may share part of the set (overlap). This idea resembles resampling methods such as cross-validation and bootstrap known in statistics for estimating the error of a predictor from limited sets of available data. In the committee framework, these techniques are recast to construct different training sets from the original training set. David Wolpert (1992) has put forward a general framework of training the committee using different partitions of the data known as stacked generalization. This approach has been adopted to the regression environment and is called stacked regression (Breiman, 1992). Stacked regression uses cross-validation to construct different sets of regression functions. A similar idea of using a bootstrap method to construct different training sets has been proposed by Breiman (1994) for classification and regression trees predictors.

## 2   THE ALGORITHMS

### 2.1   BOOTSTRAP COMMITTEE (BOOTC)

Consider a total of $N$ items are available for training. The approach is to generate $K$ replicates from the original set, each containing the same number of item as the original set. The replicates are obtained from the original set by drawing at random *with replacement*. See Efron & Tibshirani (1993) for background on bootstrapping. Use each replicate to train each network in the committee.

Using this bootstrap procedure, each replicate is expected to include roughly 36 % duplicates (due to replacement during sampling). Only the distinct fraction is used for training and the leftover fraction for early stopping, if necessary (notice slight difference from the standard bootstrapping and from Breiman's bagging). Early stopping usually requires a fraction of the data to be taken from the original training set, which might degrade the performance of the neural network. The advantage of a BOOTC is that the leftover sample is already available.

**Algorithm:**

1. Generate bootstrap replicates $L^1, \ldots, L^K$ from the original set.

2. For each bootstrap replicate, collect unsampled items into leftover sample sets, giving: $l^{*1}, \ldots, l^{*K}$.

3. For each $L^k$, train a network. Use the leftover set $l^{*k}$ as validation stopping criteria if necessary. Giving $K$ neural net predictors: $f(x; L^k)$

4. Build a committee from the bootstrap networks using a simple averaging procedure: $f_{com}(x) = \frac{1}{K} \sum_{k=1}^{K} f(x; L^k)$

There is no rule as to how many bootstrap replicates should be used to achieve a good performance. In error estimation, the number ranges from 20 to 200. It is beneficial to keep the number of replicates, hence the number of networks, small to reduce training time. Unless the networks are trained on a parallel machine, training time increases proportionally to the number of networks in the committee. In this experiment, 20 bootstrap training replicates were constructed for 20 networks in

the committee. Twenty replicates were chosen since beyond this number there is no significant improvement on the performance.

## 2.2   CROSS-VALIDATION COMMITTEE (CVC)

The algorithm is quite similar to the procedure used in prediction error estimation. First, generate replicates from the original training set by removing a fraction of the data. Let $D$ denote the original data, and $D^{-v}$ denote the data with subset $v$ removed. The procedure revolves so that each item is in the removed fraction at least once. Generate replicates $D_1^{-v1}, \ldots D_K^{-vk}$ and train each network in the committee with one replicate.

An important issue in the CVC is the degree of data overlap between the replicates. The degree of overlap depends on the number of replicates and the size of a removed fraction from the original sample. For example, if the committee consists of 5 networks and 0.5 of the data are removed for each replicate, the minimum fraction of overlap is 0 (calculation: $(v \times 2) - 1.0$) and the maximum is $\frac{4}{5}$ (calculation: $1.0 - \frac{1}{K}$).

**Algorithm:**

1. Divide data into $v$-fractions $d_1, \ldots, d_v$
2. Leave one fraction $d_k$ and train network $f_k$ with the rest of the data $(D-d_k)$.
3. Use $d_k$ as a validation stopping criteria, if necessary.
4. Build a committee from the networks using a simple averaging procedure.

The fraction of data overlap determines the trade-off between the individual network performance and error correlation between the networks. Lower correlation can be expected if the networks train with less overlapped data, which means a larger removed fraction and smaller fraction for training. The smaller the training set size, the lower the individual network performance that can be expected.

We investigated the effect of data overlap on the error correlations between the networks and the committee performance. We also studied the effect of training size on the individual performance. The goal was to find an optimal combination of data overlap and individual training size.

## 3   THE BASELINE & PERFORMANCE EVALUATION

To evaluate the improvement of the proposed methods on the committee performance, they should be compared with existing methods as the baseline. The common method for constructing a committee is to train an ensemble of networks independently. The networks in the committee are initialized with different sets of weights. This type of committee has been reported as achieving significant improvement over individual network performances in regression (Hashem, 1993) and classification tasks (Perrone, 1993; Parmanto et al., 1994).

The baseline, BOOTC, and CVC were compared using exactly the same architecture and using the same pair of training-test sets. Performance evaluation was conducted using 4-fold exhaustive cross-validation where 0.25 fraction of the original data is used for the test set and the remainder of the data is used for the training set. The procedure was repeated 4 times so that all items were once on the test set. The performance was calculated by averaging the results of 4 test sets. The simulations

were conducted several times using different initial weights to exclude the possibility that the improvement was caused by chance.

## 4 EXPERIMENTS

### 4.1 SYNTHETIC DATA: SINWAVE CLASSIFICATION

The sinwave task is a classification problem with two classes, a negative class represented as 0 and a positive class represented as 1. The data consist of two input variables, $x = (x_1, x_2)$. The entire space is divided equally into two classes with the separation line determined by the curve $x_2 = \sin(\frac{2\pi}{3}x_1)$. The upper half of the rectangle is the positive class, while the lower half is the negative one (see Fig. 1).

Gaussian noise along the perfect boundary with variance of 0.1 is introduced to the clean data and is presented in Fig. 1 (middle). Let $z$ be a vector drawn from the Gaussian distribution with variance $\eta$, then the classification rule is given by equation:

$$y(x) \begin{cases} 1 & \text{if}(x_2 + z_1) \geq \sin(\frac{2\pi}{3}(x_1 + z_2)) \\ 0 & \text{if}(x_2 + z_1) < \sin(\frac{2\pi}{3}x_1 + z_2) \end{cases} \qquad (1)$$

A similar artificial problem is used to analyze the bias-variance trade-offs by Geman et al. (1992).

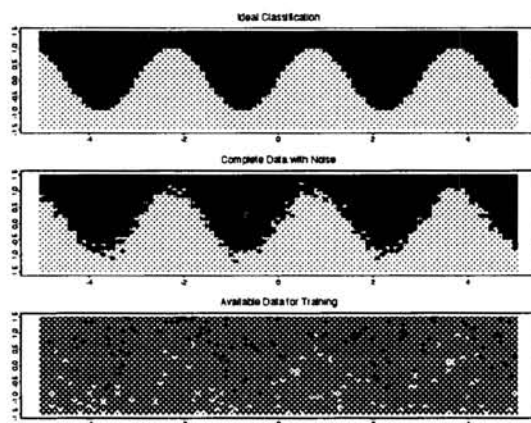

Figure 1: Complete and clean data/without noise (top), complete data with noise (middle), and a small fraction used for training (bottom).

The population contains 3030 data items, since a grid of 0.1 is used for both $x_1$ and $x_2$. In the real world, we usually have no access to the entire population. To mimic this situation, the training set contained only a small fraction of the population. Fig. 1 (bottom) visualizes a training set that contains 200 items with 100 items for each class. The training set is constructed by randomly sampling the population. The performance of the predictor is measured with respect to the test set. The population (3030 items) is used as the test set.

### 4.2 HEPATOMA DETECTION

Hepatoma is a very important clinical problem in patients who are being considered for liver transplantation for its high probability of recurrence. Early hepatoma detection may improve the ultimate outlook of the patients since special treatment can be carried out. Unfortunately, early detection using non-invasive procedures

can be difficult, especially in the presence of cirrhosis. We have been developing neural network classifiers as a detection system with minimum imaging or invasive studies (Parmanto et al., 1994).

The task is to detect the presence or absence (binary output) of a hepatoma given variables taken from an individual patient. Each data item consists of 16 variables, 7 of which are continuous variables and the rest are binary variables, primarily blood measurements.

For this experiment, 1172 data items with their associated diagnoses are available. Out of 1172 itmes, 693 items are free from missing values, 309 items contain missing values only on the categorical variables, and 170 items contain missing values on both types of variables. For this experiment, only the fraction without missing values and the fraction with missing values on the categorical variables were used, giving the total item of 1002. Out of the 1002 items, 874 have negative diagnoses and the remaining 128 have positive diagnoses.

## 4.3 BREAST CANCER

The task is to diagnose if a breast cytology is benign or malignant based on cytological characteristics. Nine input variables have been established to differentiate between the benign and malignant samples which include clump thickness, marginal adhesion, the uniformity of cell size and shape, etc.

The data set was originally obtained from the University of Wisconsin Hospitals and currently stored at the UCI repository for machine learning (Murphy & Aha, 1994). The current size of the data set is 699 examples.

## 5   THE RESULTS

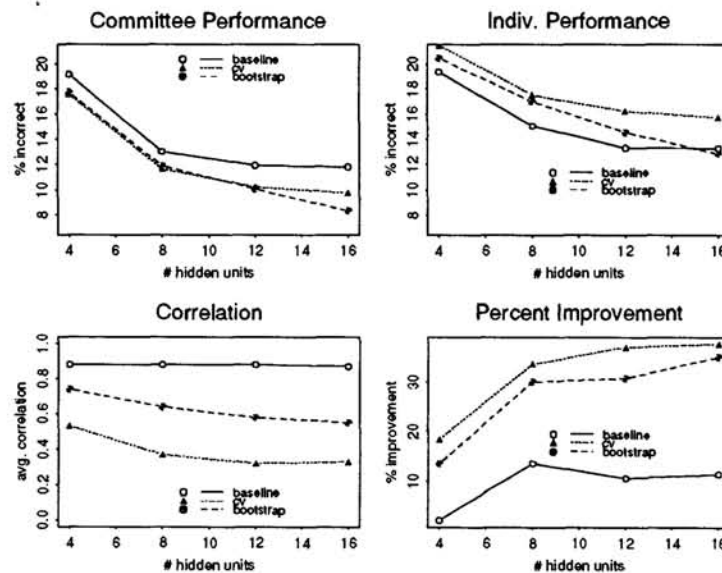

Figure 2: **Results on the sinwave classif. task.** Performances of individual nets and the committee (top); error correlation and committee improvement (bottom).

Figure 2. (top) and Table 1. show that the performance of the committee is always better than the average performance of individual networks in all three committees.

| Task | Methods | Indiv. Nets % error | Error Corr | Committee % error | Improv. to Indiv. | Improv. to baseline |
|---|---|---|---|---|---|---|
| Sinwave | Baseline | 13.31 | .87 | 11.8 | 11 % | - |
| (2 vars ) | BOOTC | 12.85 | .57 | 8.36 | 35 % | 29 % |
| | CVC | 15.72 | .33 | 9.79 | 38 % | 17 % |
| Cancer | Baseline | 2.7 | .96 | 2.5 | 5 % | - |
| (9 vars) | BOOTC | 3.14 | .83 | 2.0 | 34 % | 20 % |
| | CVC | 3.2 | .80 | 1.63 | 49 % | 35 % |
| Hepatoma | Baseline | 25.95 | .89 | 23.25 | 10.5 % | - |
| (16 vars) | BOOTC | 26.00 | .70 | 19.72 | 24 % | 15.2 % |
| | CVC | 26.90 | .55 | 19.05 | 29 % | 18 % |

Table 1: Error rate, correlation, and performance improvement *calculated based on the best architecture for each method.* Reduction of misclassification rates compare to the baseline committee

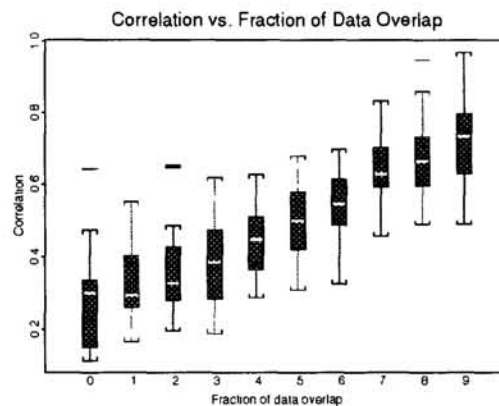

Figure 3: Error correlation and fraction of overlap in training data (results from the sinwave classification task).

The CVC and BOOTC are always better than the baseline even when the individual network performance is worse. Figure 2 (bottom) and the table show that the improvement of a committee over individual networks is proportional to the error correlation between the networks in the committee. The CVC consistently produces significant improvement over its individual network performance due to the low error correlation, while the baseline committee only produces modest improvement. This result confirms the basic assumption of this research: committee performance can be improved by decorrelating the errors made by the networks.

The performance of a committee depends on two factors: individual performance of the networks and error correlation between the networks. The gain of using BOOTC or CVC depends on how the algorithms can reduce the error correlations while still maintaining the individual performance as good as the individual performance of the baseline. The BOOTC produced impressive improvement (29 %) over the baseline on the sinwave task due to the lower correlation and good individual performance. The performances of the BOOTC on the other two tasks were not as impressive due to the modest reduction of error correlation and slight decrease in individual performance. The performances were still significantly better than the baseline committee. The CVC, on the other hand, consistently reduced the correlation and

improved the committee performance. The improvement on the sinwave task was not as good as the BOOTC due to the low individual performance.

The individual performance of the CVC and BOOTC in general are worse than the baseline. The individual performance of CVC is 18 % and 19 % lower than the baseline on the sinwave and cancer tasks respectively, while the BOOTC suffered significant reduction of individual performance only on the cancer task (16 %). The degradation of individual performance is due to the smaller training set for each network on the CVC and the BOOTC. The detrimental effect of a small training set, however, is compensated by low correlation between the networks. The effect of a smaller training set depends on the size of the original training set. If the data size is large, using a smaller set may not be harmful. On the contrary, if the data set is small, using an even smaller data set can significantly degrade the performance.

Another interesting finding of this experiment is the relationship between the error correlation and the overlap fraction in the training set. Figure 3 shows that small data overlap causes the networks to have low correlation to each other.

## 6   SUMMARY

Training committees of networks using different set of data resampled from the original training set can improve committee performance by reducing the error correlation among the networks in the committee. Even when the individual network performances of the BOOTC and CVC degrade from the baseline networks, the committee performance is still better due to the lower correlation.

**Acknowledgement**

This study is supported in part by Project Grant DK 29961 from the National Institutes of Health, Bethesda, MD. We would like to thank the Pittsburgh Transplantation Institute for providing the data for this study.

**References**

Breiman, L, (1992) *Stacked Regressions*, TR 367, Dept. of Statistics., UC. Berkeley.

Breiman, L, (1994) *Bagging Predictors*, TR 421, Dept. of Statistics, UC. Berkeley.

Efron, B., & Tibshirani, R.J. (1993) *An Introd. to the Bootstrap*. Chapman & Hall.

Hashem, S. (1994). *Optimal Linear Combinations of Neural Networks*. PhD Thesis, Purdue University.

Geman, S., Bienenstock, E., and Doursat, R. (1992) Neural networks and the bias/variance dilemma. *Neural Computation*, 4(1), 1-58.

Murphy, P. M., & Aha, D. W. (1994). *UCI Repository of machine learning databases* [ftp: ics.uci.edu/pub/machine-learning-databases/]

Parmanto, B., Munro, P.W., Doyle, H.R., Doria, C., Aldrighetti, L., Marino, I.R., Mitchel, S., and Fung, J.J. (1994) Neural network classifier for hepatoma detection. *Proceedings of the World Congress of Neural Networks 1994* San Diego, June 4-9.

Perrone, M.P. (1993) *Improving Regression Estimation: Averaging Methods for Variance Reduction with Extension to General Convex Measure Optimization*. PhD Thesis, Department of Physics, Brown University.

Wolpert, D. (1992). Stacked generalization, *Neural Networks*, 5, 241-259.